# Bayesian Warped Gaussian Processes

**Miguel Lázaro-Gredilla**
Dept. Signal Processing & Communications
Universidad Carlos III de Madrid - Spain
`miguel@tsc.uc3m.es`

## Abstract

Warped Gaussian processes (WGP) [1] model output observations in regression tasks as a parametric nonlinear transformation of a Gaussian process (GP). The use of this nonlinear transformation, which is included as part of the probabilistic model, was shown to enhance performance by providing a better prior model on several data sets. In order to learn its parameters, maximum likelihood was used. In this work we show that it is possible to use a non-parametric nonlinear transformation in WGP and variationally integrate it out. The resulting Bayesian WGP is then able to work in scenarios in which the maximum likelihood WGP failed: Low data regime, data with censored values, classification, etc. We demonstrate the superior performance of Bayesian warped GPs on several real data sets.

## 1 Introduction

In a Bayesian setting, the Gaussian process (GP) is commonly used to define a prior probability distribution over functions. This leads to a simple and elegant probabilistic framework that allows to solve, among others, regression and classification tasks, achieving state-of-the-art performance [2, 3]. For a thorough treatment on GPs, the reader is referred to [4].

In the regression setting, output data are often modelled directly as observations from a GP. However, it is shown in [1] that for some data sets, better models can be built if the observed outputs are regarded as a nonlinear distortion (the so-called *warping*) of a GP instead. For a warped GP (WGP), the warping function can take any parametric form, and in [1] the sum of a linear function and several `tanh` functions is used. The parameters defining the transformation are then learned using maximum likelihood. WGPs have the advantage of having a closed-form expression for the evidence and have been applied in a number of works [5, 6], but also have several shortcomings: Maximum likelihood learning might result in overfitting if a warping function with too many parameters is used (or if too few data are available), it does not model additional output noise after the warping, it cannot model "flat" warping functions for reasons explained below and, as a consequence, runs into problems when observations are clustered (many output data take the same value). In this work we set out to show that it is possible to place another GP prior on the warping function and variationally integrate it out. By doing so, all of the aforementioned problems disappear and we can enjoy the benefits of WGPs on a wider selection of scenarios.

The remainder of this work is organised as follows: In Section 2 we introduce the Bayesian WGP model, which is analytically intractable. In Section 3, a variational lower bound on the exact evidence of the model is derived, which allows for approximate inference and hyperparameter learning. We show the advantages of integrating out the warping function in Section 4, where we compare the performance of the maximum likelihood and the Bayesian versions of warped GPs. Finally, we wrap-up with some concluding remarks in Section 5.

## 2 The Bayesian warped Gaussian process model

Given a set of input values $\{\mathbf{x}_i \in \mathbb{R}^D\}_{i=1}^n$ and their associated targets $\{\mathbf{y}_i \in \mathbb{R}\}_{i=1}^n$, we define the Bayesian warped Gaussian process (BWGP) model as

$$y_i = g(f(\mathbf{x}_i)) + \varepsilon_i \tag{1a}$$

where $f(\mathbf{x})$ is a (possibly noisy) latent function with $D$-dimensional inputs, $g(f)$ is an arbitrary warping function with scalar inputs and $\varepsilon$ is a Gaussian noise term. Proceeding in a Bayesian fashion, we place priors on $g$, $f$, and $\varepsilon_i$. We use Gaussian process and normal priors

$$f(\mathbf{x}) \sim \mathcal{GP}(\mu_0, k(\mathbf{x}, \mathbf{x}')), \qquad g(f) \sim \mathcal{GP}(f, c(f, f')), \qquad \varepsilon_i \sim \mathcal{N}(0, \sigma^2). \tag{1b}$$

Notice that by setting the prior mean on $g(f)$ to $f$, we assume that the warping is "by default" the identity. For $f$, any valid covariance function $k(\mathbf{x}, \mathbf{x}')$ can be used, whereas for the warping function $g$ we use a squared exponential: $c(f, f') = \sigma_g^2 \exp(-(f - f')^2/(2\ell^2))$. The mentioned hyperparameters as well as those included in $k(\mathbf{x}, \mathbf{x}')$ can be collected in $\boldsymbol{\theta} \equiv \{\boldsymbol{\theta}_k, \sigma_g, \ell, \sigma, \mu_0\}$.

It might seem that since $f(\mathbf{x})$ is already an arbitrary nonlinear function, further distorting its output through $g(f)$ is an additional unnecessary complication. However, even though $g(f(\mathbf{x}))$ can model arbitrary functions just as $f(\mathbf{x})$ is able to, the implied *prior* is very different since the composition of two GPs $g(f(\mathbf{x}))$ is no longer a GP. This is the same idea as with copulas, but here the warping function $g(f)$ is treated in non-parametric form.

### 2.1 Relationship with maximum likelihood warped Gaussian processes

Though the idea of distorting a standard GP is common to WGP and BWGP, there are several relevant differences worth clarifying:

In [1], noise is present only in latent function $f(\mathbf{x})$ and observed data corresponds exactly to the warping of $f(\mathbf{x})$. BWGP has an additional noise term $\varepsilon$ that can account for extra noise in the observations after warping. This term can be neglected by setting $\sigma^2 = 0$.

BWGP places a prior on the warping function, instead of using a parametric definition, which allows for maximum flexibility while avoiding overfitting. On the other hand, by choosing the number of $\tanh$ functions in their parametric warping function, WGP sets a trade-off between both.

Finally, the definition of the warping function is reversed between BWGP and WGP. If no noise is present, our warping function $y = g(f)$ maps latent space $f$ to output space $y$. In contrast, in [1] the inverse mapping $f = w(y)$ is defined due to analytical tractability reasons. Because of this, the warping function in [1] is restricted to be monotonic, so that it is possible to unambiguously identify its inverse $y = w^{-1}(f) = g(f)$ and thus define a valid probability distribution in output space. Since we already work with the direct warping function $g(f)$, we do not need to impose any constraint on it and thus can use a GP prior. Also, as discussed in [1], WGPs cannot deal properly with models that involve a "flat" region (i.e., $g'(f) = 0$) in the warping function (such as ordinal regression or classification), since the inverse $w(y) = g^{-1}(y)$ is not well defined. These flat regions result in probability *masses* in output space. In those cases, the probability *density* of data under the WGP model (the evidence) will be infinity, so that it cannot be used for model selection and numerical computation becomes unstable. None of this problems arise on BWGP, which can handle both continuous and discrete observations and model warping functions with flat regions.

### 2.2 Relationship with other Gaussian processes models

For a given warping function $g(f)$, BWGP can be seen as a standard GP model with likelihood $p(y_i|f(\mathbf{x}_i)) = \mathcal{N}(y_i|g(f(\mathbf{x}_i)), \sigma^2)$. Different choices for $g(f)$ result in different GP models:

- GP regression [7]: Corresponds to setting $g(f) = f$ (the mean in our prior).
- GP classification [3]: Corresponds to setting $g(f) = \text{sign}(f)$ with $y_i \in \{-1, +1\}$ and $\sigma^2 = 0$. Using a noisy latent function $f(\mathbf{x})$ as prior and a step function as likelihood is equivalent to using a noiseless latent function as prior and normal cdf sigmoid function as likelihood [4], so this model corresponds exactly with GP probit classification.

- Ordinal (noisy) regression [8]: Corresponds to setting $g(f) = \sum_{k=1}^{K} H(f - b_k)$ and optionally setting $\sigma^2 = 0$. $H(f)$ is the Heaviside step function and $b_k$ are parameters defining the widths and locations of the $K$ bins in latent space.

- Maximum likelihood WGP [1]: Corresponds to setting $g(f) = w^{-1}(f)$ and $\sigma^2 = 0$.

Because $g(f)$ is integrated out, all of the above models, and possibly many others, can be learned using BWGP. We will see examples of problems requiring other likelihoods in Section 4. Thus, to some extent, BWGP can be regarded as *likelihood learning* tool.

## 3 Variational inference for BWGP

Analytical inference in the BWGP model (1) is intractable. Instead of resorting to expensive Monte Carlo methods, we will develop an efficient variational approximation of comparable computational cost to that of WGP. We follow ideas discussed in [9] in order to gain tractability.

### 3.1 Augmented model

First, let us rewrite (1) instantiated only at the available observations $\mathbf{y} = [y_1 \ldots y_n]^\top$. We omit conditioning on inputs $\{\mathbf{x}_i\}_{i=1}^n$ and hyperparameters $\boldsymbol{\theta}$. We have

$$p(\mathbf{y}|\mathbf{g}) = \mathcal{N}(\mathbf{y}|\mathbf{g}, \sigma^2 \mathbf{I}) \quad p(\mathbf{g}|\mathbf{f}) = \mathcal{N}(\mathbf{g}|\mathbf{f}, \mathbf{C}_{ff}) \quad p(\mathbf{f}) = \mathcal{N}(\mathbf{f}|\boldsymbol{\mu_0}, \mathbf{K}), \quad (2)$$

where $\mathbf{f} = [f_1 \ldots f_n]^\top$ is the latent function evaluated at the training inputs $\{\mathbf{x}_1 \ldots \mathbf{x}_n\}$ and $\mathbf{g} = [g_1 \ldots g_n]^\top$ is the warping function evaluated at $\mathbf{f}$. We use $\mathbf{K}$ to refer to the $n \times n$ covariance matrix of the latent function, with entries $[\mathbf{K}]_{ij} = k(\mathbf{x}_i, \mathbf{x}_j)$, whereas similarly $[\mathbf{C}_{ff}]_{ij} = c(f_i, f_j)$ is the $n \times n$ warping covariance matrix. In general, we use $[\mathbf{C}_{ab}]_{ij} = c(a_i, b_j)$.

Now we proceed as in sparse GPs [10] and augment this model with a set of $m$ inducing variables $\mathbf{u} = [u_1 \ldots u_m]^\top$ that correspond to evaluating function $u(v) = g(v) - v$ at some auxiliary values $v_1 \ldots v_m$. We can expand $p(\mathbf{g}|\mathbf{f})$ by first conditioning on $\mathbf{u}$ to obtain $p(\mathbf{g}|\mathbf{u}, \mathbf{f})$, and then including the prior $p(\mathbf{u})$. This yields the augmented model

$$p(\mathbf{y}|\mathbf{g}) = \mathcal{N}(\mathbf{y}|\mathbf{g}, \sigma^2 \mathbf{I}) \qquad p(\mathbf{g}|\mathbf{u}, \mathbf{f}) = \mathcal{N}(\mathbf{g}|\mathbf{f} + \mathbf{C}_{fv}\mathbf{C}_{vv}^{-1}\mathbf{u}, \mathbf{C}_{ff} - \mathbf{C}_{fv}\mathbf{C}_{vv}^{-1}\mathbf{C}_{fv}^\top) \quad (3a)$$

$$p(\mathbf{u}) = \mathcal{N}(\mathbf{u}|\mathbf{0}, \mathbf{C}_{vv}) \qquad\qquad p(\mathbf{f}) = \mathcal{N}(\mathbf{f}|\mathbf{0}, \mathbf{K}) \quad (3b)$$

Note that the original model (2) and the augmented model (3) are exactly identical, since we can marginalise $\mathbf{u}$ out from (3) to get exactly (2). In other words, we introduced $\mathbf{u}$ in a consistent manner, so that $\int p(\mathbf{g}|\mathbf{u}, \mathbf{f})p(\mathbf{u})d\mathbf{u} = p(\mathbf{g}|\mathbf{f})$. The inclusion of the inducing variables does not change the model, independently of their number $m$ or their locations $v_1 \ldots v_m$.

Inducing variables $\mathbf{u}$ have a physical interpretation in this model. Expressing the warping function as $g(v) = u(v) + v$, the inducing variables correspond to evaluating GP $u(v)$ at locations $v_1 \ldots v_m$, which live in latent space (just as $f$ does). Observe that $\mathbf{u}$ provides a probabilistic description of the warping function. In particular, as $m$ grows and the sampling in latent space becomes more and more dense[1], the covariance $\mathbf{C}_{ff} - \mathbf{C}_{fv}\mathbf{C}_{vv}^{-1}\mathbf{C}_{fv}^\top$ gets closer to zero[2] and $p(\mathbf{g}|\mathbf{u}, \mathbf{f})$ becomes a Dirac delta, thus making the warping function deterministic given $\mathbf{u}$, $g(f) = f + [c(f, v_1) \ldots c(f, v_m)]\mathbf{C}_{vv}^{-1}\mathbf{u}$.

### 3.2 Variational lower bound

The exact posterior of BWGP model (3) is analytically intractable. We can proceed by selecting, within a given family of distributions, the approximate posterior $q(\mathbf{g}, \mathbf{u}, \mathbf{f})$ that minimises the Kullback-Leibler (KL) divergence to the true posterior $p(\mathbf{g}, \mathbf{u}, \mathbf{f}|\mathbf{y})$. We can write

$$\log p(\mathbf{y}) \geq \log p(\mathbf{y}) - \mathrm{KL}(q(\mathbf{g}, \mathbf{u}, \mathbf{f})||p(\mathbf{g}, \mathbf{u}, \mathbf{f}|\mathbf{y})) = \int q(\mathbf{g}, \mathbf{u}, \mathbf{f}) \log \frac{p(\mathbf{y}, \mathbf{g}, \mathbf{u}, \mathbf{f})}{q(\mathbf{g}, \mathbf{u}, \mathbf{f})} \mathrm{d}\mathbf{g}\mathrm{d}\mathbf{f}\mathrm{d}\mathbf{u} = \mathcal{F},$$

where $\mathcal{F}$ is a variational lower bound on the evidence $\log p(\mathbf{y})$. Since $\log p(\mathbf{y})$ is constant for any choice of $q$, it is obvious that maximising $\mathcal{F}$ wrt $q$ yields the best approximation in the mentioned KL sense within the considered family of distributions. We should choose a family that can model the posterior as well as possible while keeping the computation of $\mathcal{F}$ tractable. If no constraints on $q$ are imposed, maximisation retrieves the exact posterior.

We expand $q(\mathbf{g}, \mathbf{u}, \mathbf{f}) = q(\mathbf{g}|\mathbf{u}, \mathbf{f})q(\mathbf{u}|\mathbf{f})q(\mathbf{f})$ and constrain it as follows: $q(\mathbf{f}) = \mathcal{N}(\mathbf{f}|\boldsymbol{\mu}, \boldsymbol{\Sigma})$, $q(\mathbf{u}|\mathbf{f}) = q(\mathbf{u})$, $q(\mathbf{g}|\mathbf{u}, \mathbf{f}) = p(\mathbf{g}|\mathbf{u}, \mathbf{f})$. We argue that this constraints should still allow for a good approximation: The exact posterior over $\mathbf{f}$ for any monotonic warping function is Gaussian (see [1]), so it is reasonable to set $q(\mathbf{f})$ to be a Gaussian; GPs $u(v)$ and $f(\mathbf{x})$ are independent a priori and encode different parts of the model, so it is reasonable to approximate them as independent a posteriori $q(\mathbf{u}|\mathbf{f}) = q(\mathbf{u})$; and finally, given a dense sampling of the latent space (which is feasible, since it is one-dimensional), $p(\mathbf{g}|\mathbf{u}, \mathbf{f})$ is virtually a Dirac delta, so conditioning on the observations has no effect and we can set $q(\mathbf{g}|\mathbf{u}, \mathbf{f}) = p(\mathbf{g}|\mathbf{u}, \mathbf{f})$. Using the constrained expansion for $q$ we get

$$\mathcal{F}(q(\mathbf{u}), \boldsymbol{\mu}, \boldsymbol{\Sigma}) = \int q(\mathbf{u})q(\mathbf{f}) \left( \int p(\mathbf{g}|\mathbf{u}, \mathbf{f}) \log p(\mathbf{y}|\mathbf{g}) \mathrm{dg} + \log \frac{p(\mathbf{u})}{q(\mathbf{u})} \right) \mathrm{dfdu} - \mathrm{KL}(q(\mathbf{f})||p(\mathbf{f}))$$

The inner integral yields

$$\int p(\mathbf{g}|\mathbf{u}, \mathbf{f}) \log p(\mathbf{y}|\mathbf{g}) \mathrm{dg} = -\frac{n}{2} \log(2\pi\sigma^2) - \frac{1}{2\sigma^2} \{ \mathrm{trace}(\mathbf{C}_{ff} - \mathbf{C}_{fv}\mathbf{C}_{vv}^{-1}\mathbf{C}_{fv}^{\top}) + ||\mathbf{y} - \mathbf{f}||^2$$
$$- 2\mathbf{y}^{\top}\mathbf{C}_{fv}\mathbf{C}_{vv}^{-1}\mathbf{u} + \mathbf{u}^{\top}\mathbf{C}_{vv}^{-1}\mathbf{C}_{fv}^{\top}\mathbf{C}_{fv}\mathbf{C}_{vv}^{-1}\mathbf{u} + 2\mathbf{u}^{\top}\mathbf{C}_{vv}^{-1}\mathbf{C}_{fv}^{\top}\mathbf{f} \},$$

which can be averaged analytically over $q(\mathbf{f}) = \mathcal{N}(\mathbf{f}|\boldsymbol{\mu}, \boldsymbol{\Sigma})$. To this end, we define $\psi_0 = \langle \mathrm{trace}(\mathbf{C}_{ff}) \rangle_{q(\mathbf{f})}$, $\boldsymbol{\Psi}_2 = \langle \mathbf{C}_{fv}^{\top}\mathbf{C}_{fv} \rangle_{q(\mathbf{f})}$, $\boldsymbol{\Psi}_1 = \langle \mathbf{C}_{fv} \rangle_{q(\mathbf{f})}$, and $\boldsymbol{\psi}_3 = \langle \mathbf{C}_{fv}^{\top}\mathbf{f} \rangle_{q(\mathbf{f})}$, which are

$$\psi_0 = n\sigma_g^2 \qquad\qquad [\boldsymbol{\Psi}_2]_{jk} = \sum_{i=1}^{n} \frac{\sigma_g^4 \ell \exp\left(-\frac{(v_j - v_k)^2}{4\ell^2} - \frac{([\boldsymbol{\mu}]_i - (v_j + v_k)/2)^2}{2[\boldsymbol{\Sigma}]_{ii} + \ell^2}\right)}{\sqrt{2[\boldsymbol{\Sigma}]_{ii} + \ell^2}}$$

$$[\boldsymbol{\Psi}_1]_{ij} = \frac{\sigma_g^2 \ell \exp\left(-\frac{([\boldsymbol{\mu}]_i - v_j)^2}{2([\boldsymbol{\Sigma}]_{ii} + \ell^2)}\right)}{\sqrt{[\boldsymbol{\Sigma}]_{ii} + \ell^2}} \quad [\boldsymbol{\psi}_3]_j = \sum_{i=1}^{n} \frac{\sigma_g^2 \ell \exp\left(-\frac{([\boldsymbol{\mu}]_i - v_j)^2}{2([\boldsymbol{\Sigma}]_{ii} + \ell^2)}\right)([\boldsymbol{\mu}]_i \ell^2 - [\boldsymbol{\Sigma}]_{ii} v_j)}{\sqrt{([\boldsymbol{\Sigma}]_{ii} + \ell^2)^3}}.$$

After averaging over $q(\mathbf{f})$, most of the terms do not depend on $\mathbf{u}$ and can be taken out of the integral. The remaining terms which depend on $\mathbf{u}$ can be arranged as follows:

$$\int q(\mathbf{u}) \log \frac{p(\mathbf{u}) \exp(-\frac{1}{2\sigma^2}\mathbf{u}^{\top}\mathbf{C}_{vv}^{-1}\boldsymbol{\Psi}_2\mathbf{C}_{vv}^{-1}\mathbf{u} + \frac{1}{\sigma^2}(\mathbf{y}^{\top}\boldsymbol{\Psi}_1 - \boldsymbol{\psi}_3^{\top})\mathbf{C}_{vv}^{-1}\mathbf{u})}{q(\mathbf{u})} \mathrm{du}. \qquad (4)$$

Note that we have not specified any functional form for $q(\mathbf{u})$, so any distribution over $\mathbf{u}$ is valid. In particular, we want to choose $q(\mathbf{u})$ so as to maximise (4), because that would be the choice that maximises $\mathcal{F}(q(\mathbf{u}), \boldsymbol{\mu}, \boldsymbol{\Sigma})$. Inspecting (4), we notice that it has the form of a Jensen's inequality lower bound. The maximum wrt $q(\mathbf{u})$ can then be obtained by reversing Jensen's inequality:

$$\log \int p(\mathbf{u}) \exp\left(-\frac{1}{2\sigma^2}\mathbf{u}^{\top}\mathbf{C}_{vv}^{-1}\boldsymbol{\Psi}_2\mathbf{C}_{vv}^{-1}\mathbf{u} + \frac{1}{\sigma^2}(\mathbf{y}^{\top}\boldsymbol{\Psi}_1 - \boldsymbol{\psi}_3^{\top})\mathbf{C}_{vv}^{-1}\mathbf{u}\right) \mathrm{du}$$

$$= \frac{1}{2\sigma^2}(\mathbf{y}^{\top}\boldsymbol{\Psi}_1 - \boldsymbol{\psi}_3^{\top})(\boldsymbol{\Psi}_2 + \sigma^2\mathbf{C}_{vv})^{-1}(\boldsymbol{\Psi}_1^{\top}\mathbf{y} - \boldsymbol{\psi}_3) - \frac{1}{2} \log \frac{|\boldsymbol{\Psi}_2 + \sigma^2\mathbf{C}_{vv}|}{|\mathbf{C}_{vv}|} + \frac{n}{2} \log \sigma^2,$$

which corresponds to selecting[3] $q^*(\mathbf{u}) = \mathcal{N}(\mathbf{u} \mid \mathbf{C}_{vv}\boldsymbol{\beta}, \ \sigma^2\mathbf{C}_{vv}(\boldsymbol{\Psi}_2 + \sigma^2\mathbf{C}_{vv})^{-1}\mathbf{C}_{vv})$ with $\boldsymbol{\beta} = (\boldsymbol{\Psi}_2 + \sigma^2\mathbf{C}_{vv})^{-1}(\boldsymbol{\Psi}_1^{\top}\mathbf{y} - \boldsymbol{\psi}_3)$. Replacing one of the variational distributions within the bound by its optimal value is sometimes referred to as using a "marginalised variational bound" [11]. Grouping all terms together, we finally obtain:

$$\mathcal{F}_{\mathrm{BWGP}}(\boldsymbol{\mu}, \boldsymbol{\Sigma}) = -\frac{1}{2\sigma^2}(||\mathbf{y} - \boldsymbol{\mu}||^2 + \mathrm{trace}(\boldsymbol{\Sigma}) + \psi_0 - \mathrm{trace}(\boldsymbol{\Psi}_2\mathbf{C}_{vv}^{-1})) - \frac{1}{2} \log \frac{|\boldsymbol{\Psi}_2 + \sigma^2\mathbf{C}_{vv}|}{|\mathbf{C}_{vv}|}$$

$$+ \frac{1}{2\sigma^2}(\mathbf{y}^{\top}\boldsymbol{\Psi}_1 - \boldsymbol{\psi}_3^{\top})(\boldsymbol{\Psi}_2 + \sigma^2\mathbf{C}_{vv})^{-1}(\boldsymbol{\Psi}_1^{\top}\mathbf{y} - \boldsymbol{\psi}_3) - \frac{n}{2} \log 2\pi - \mathrm{KL}(\mathcal{N}(\boldsymbol{\mu}, \boldsymbol{\Sigma})||\mathcal{N}(\mu_0, \mathbf{K}))$$

This bound depends on $\boldsymbol{\mu}$ and $\boldsymbol{\Sigma}$, i.e., $n+n(n+1)/2$ variational parameters which must be optimised. Even for moderate sizes of $n$, this can be inconvenient. Following [12, 13], we can reduce the number of free parameters by considering the conditions that must be met at any local maxima. By imposing $\frac{\partial \mathcal{F}(\boldsymbol{\mu},\boldsymbol{\Sigma})}{\partial \boldsymbol{\Sigma}} = 0$, we know that the posterior covariance can be expressed as $\boldsymbol{\Sigma} = (\mathbf{K}^{-1} + \boldsymbol{\Lambda})^{-1}$, for some diagonal matrix $\boldsymbol{\Lambda}$. With this definition, the bound $\mathcal{F}(\boldsymbol{\mu}, \boldsymbol{\Lambda})$ now depends only on $2n$ free variational parameters and can be computed in $\mathcal{O}(n^3)$ time and $\mathcal{O}(n^2)$ space, just as WGP.

### 3.3 Model selection

The gradients of the variational bound $\mathcal{F}_{\text{BWGP}}(\boldsymbol{\mu}, \boldsymbol{\Lambda}, \boldsymbol{\theta})$ (now explicitly including its dependence on the hyperparameters) can be computed analytically so it is possible to jointly optimise it both wrt to the $2n$ free variational parameters and hyperparameters $\boldsymbol{\theta}$ in order to simultaneously perform model selection (by choosing the hyperparameters) and obtaining an accurate posterior (by choosing the free variational parameters). The hyperparameters are the same as for a WGP that uses *a single* tanh function, so no overfitting is expected, while still enjoying a completely flexible warping function.

### 3.4 Approximate predictive density

In order to use the proposed approximate posterior to make predictions for a new test output $y_*$ given input $\mathbf{x}_*$ we need to compute $q(y_*|\mathbf{y}) = \int p(y_*|g_*)p(g_*|f_*, \mathbf{u})q(\mathbf{u})p(f_*|\mathbf{f})q(\mathbf{f})\mathrm{d}g_*\mathrm{d}f_*\mathrm{d}\mathbf{u}\mathrm{d}\mathbf{f}$. Integration wrt all variables can be computed analytically except for $f_*$, resulting in

$$q(y_*|\mathbf{y}) = \int q(y_*|f_*)q(f_*|\mathbf{y})\mathrm{d}f_*$$

with $q(y_*|f_*) = \mathcal{N}(y_* \mid f_* + \mathbf{c}_*^\top\boldsymbol{\beta}, \ \sigma^2 + c_{**} - \mathbf{c}_*^\top(\mathbf{C}_{vv} + \sigma^2\mathbf{C}_{vv}\boldsymbol{\Psi}_2^{-1}\mathbf{C}_{vv})^{-1}\mathbf{c}_*)$ and $q(f_*|\mathbf{y}) = \mathcal{N}(f_*|\mu_*, \sigma_*^2)$, where $\mu_* = \mu_0 + \mathbf{k}_*^\top\mathbf{K}^{-1}(\boldsymbol{\mu} - \mu_0\mathbf{1})$, $\sigma_*^2 = k_{**} - \mathbf{k}_*^\top(\mathbf{K} + \boldsymbol{\Lambda}^{-1})^{-1}\mathbf{k}_*$, $\mathbf{k}_* = [k(\mathbf{x}_*, \mathbf{x}_1)\dots k(\mathbf{x}_*, \mathbf{x}_n)]^\top$, $k_{**} = k(\mathbf{x}_*, \mathbf{x}_*)$, $\mathbf{c}_* = [c(f_*, f_1)\dots c(f_*, f_n)]^\top$, $c_{**} = c(f_*, f_*) = \sigma_g^2$ and $\mathbf{1}$ is an appropriately sized vector of ones.

This latter one-dimensional integral can be computed numerically if needed, using Gaussian quadrature techniques. However, the posterior mean and variance can be computed analytically. Indeed,

$$\mathbb{E}_q[y_*|\mathbf{y}] = \mu_* + \boldsymbol{\Psi}_{1*}\boldsymbol{\beta} \qquad \mathbb{V}_q[y_*|\mathbf{y}] = \boldsymbol{\beta}^\top(\boldsymbol{\Psi}_{2*} - \boldsymbol{\Psi}_{1*}^\top\boldsymbol{\Psi}_{1*})\boldsymbol{\beta} + 2(\boldsymbol{\psi}_{3*}^\top - \mu_*^\top\boldsymbol{\Psi}_{1*})\boldsymbol{\beta} + \sigma_*^2$$
$$+ \sigma^2 + c_{**} - \text{trace}(\boldsymbol{\Psi}_{2*}(\mathbf{C}_{vv} + \sigma^2\mathbf{C}_{vv}\boldsymbol{\Psi}_2^{-1}\mathbf{C}_{vv})^{-1})$$

where $\boldsymbol{\Psi}_*$ matrices are defined as their non-starred counterparts, but using $\mu_*$ and $\sigma_*^2$ instead of $\boldsymbol{\mu}$ and $\boldsymbol{\Sigma}$ in their computation. In spite of this, the approximate posterior is *not* Gaussian in general.

## 4 Experiments

We will now investigate the behaviour of BWGP on several real regression and classification datasets. In our experiments we will compare its performance with that of the original implementation[4] of the maximum likelihood WGP model from [1]. In order to show the effect of varying the complexity of the parametric warping function in WGP, we tested a 3 tanh model (the default, used in the experiments from [1]) and a 20 tanh model, denoted as WGP3 and WGP20, respectively. We did our best to achieve the maximum accuracy in WGP, so in order to solve each data split, we optimised its hyperparameters 5 times from a random initialisation (the implementation's default method) and 5 times more using a standard GP to initialise the underlying GP (and randomly initialising the warping function). Out of the 10 total runs, we used the one achieving a higher evidence. The BWGP model was initialised from a standard GP and run only once per data split. The standard ARD SE covariance function [4] plus noise was used for the underlying GP in all models. The two measures that we use to compare performance are MSE $= \frac{1}{n_*}\sum_{i=1}^{n_*}(y_{*i} - \mathbb{E}_q[y_{*i}|\mathbf{y}])^2$ and NLPD $= -\frac{1}{n_*}\sum_{i=1}^{n_*}\log q(y_{*i}|\mathbf{y})$. In both cases, a lower value indicates better performance.

### 4.1 Toy 1D data

First we evaluate the model on a simple one-dimensional toy problem. In order to generate a non-linearly distorted signal, we round a sine function to the nearest integer and add Gaussian noise with variance $\sigma^2 = 2.5 \times 10^{-3}$. The training set consists of 51 uniformly spaced samples between $-\pi$ and $\pi$. We train a standard GP, WGP, and BWGP and then we test them on 401 uniformly spaced samples in the same interval. Results are displayed on Fig. 1.

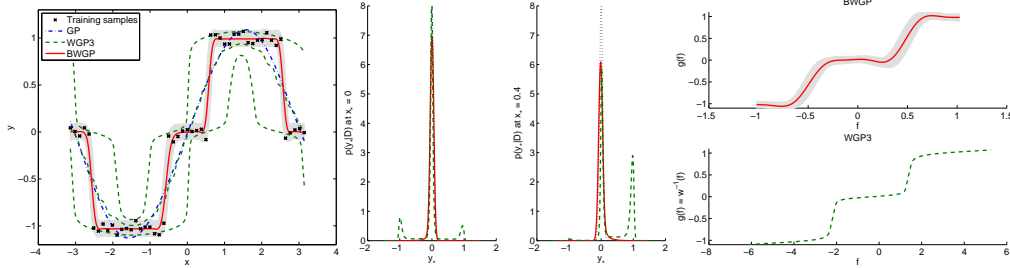

Figure 1: Left: Posterior mean for the proposed models. A dashed envelope encloses 90% posterior mass for WGP, whereas a shading is used to show 90% posterior mass for BWGP. Middle: The dotted line shows the true posterior at $x = 0$ and $x = 0.4$, which is much better modelled by BWGP. Right: Warping functions inferred by each model.

The warping functions look reasonable for both models. For WGP it is a deterministic function, the inverse of the strictly monotonic function $w(y)$, so it can never achieve completely "flat" zones. Since WGP does not model output noise explicitly, these flat zones transfer and magnify output noise to latent space, with the consequent degradation in performance. Note the extra spread of the posterior mass in comparison with the actual training data, which is much better modelled by BWGP. The mean of WGP fails to follow the flat regions at zero, behaving as a sine function, just like the standard GP. The standard GP is also unable to handle this signal properly because of the non-stationary smoothness: Abrupt changes are followed by constant levels. BWGP is able to deal properly with noisy quantised signals and it is able to learn the implicit quantisation function.

### 4.2 Regression data sets

We now turn to the three real data sets originally used in [1] to assess WGP and for which it is specially suited. These are: abalone [14] (4177 samples, 8 dimensions), creep [15, 16] (2066 samples, 30 dimensions), and ailerons [17] (7154 samples, 40 dimensions). As for the size of the training set, the typical choice is to use 1000, 800 and 1000 samples respectively. For each problem, we generated 60 splits by randomly partitioning data. Results are displayed on Table 1. The warping functions inferred by BWGP are displayed in Fig. 3(a)-(c) and are almost identical to those displayed in [1] for WGP. The shading represents 99.99% posterior mass.

Table 1: NMSE and NLPD figures for the compared methods on original data sets of [1].

| Model | MSE | | | NLPD | | |
|---|---|---|---|---|---|---|
| | abalone | creep | ail $_{(\times 10^{-8})}$ | abalone | creep | ailerons |
| GP | 4.55±0.14 | 584.9±71.2 | 2.95±0.16 | 2.17±0.01 | 4.46±0.03 | -7.30±0.01 |
| BWGP | 4.55±0.11 | 491.8±36.2 | 2.91±0.14 | 1.99±0.01 | 4.31±0.04 | -7.38±0.02 |
| MLWGP3 | 4.54±0.10 | 502.3±43.3 | 2.80±0.11 | 1.97±0.02 | 4.21±0.03 | -7.44±0.01 |
| MLWGP20 | 4.59±0.32 | 506.3±46.1 | 3.42±2.87 | 1.99±0.05 | 4.21±0.08 | -7.45±0.08 |

In terms of NLPD, BWGP always outperforms the standard GP, but it is in turn outperformed by the maximum likelihood variants, which do not need to resort to any approximation to compute its posterior. In terms of MSE, BWGP always performs better than WGP20 on these data sets, but only performs better than WGP3 on the creep data set, which, on the other hand, is the one that seems

to benefit more from the use of a warping function. It seems that the additional flexibility of the warping function in WGP20 is penalising its ability to generalise properly.

Upon seeing these results, we can conclude that WGP3 is already a good enough solution when abundant training data are available and a simple warping function is required. This is reasonable: The additional number of hyperparameters is small (only 9) and inference can be performed analytically. We can also see in Fig. 3(a)-(c) that the posterior over the warping functions is highly peaked, so a maximum likelihood approach makes sense. However, performance might suffer when the warping function becomes even slightly complex, as in `creep`, or when the number of available data for training is very small (see the effect of the training set size on Fig. 2). In those cases, BWGP is a safer option, since it will not overfit independently of the amount of data while allowing for a highly flexible warping function.

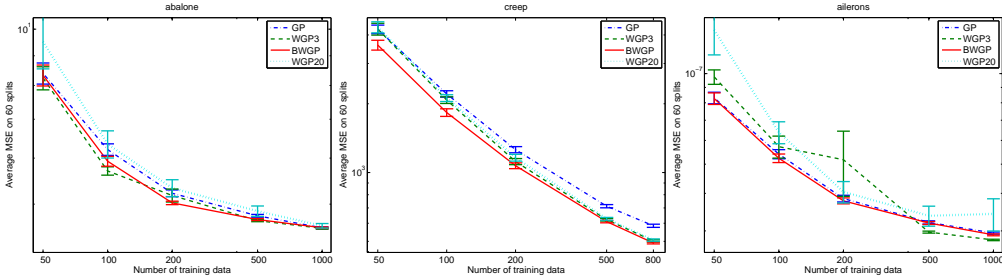

Figure 2: Average MSE, as well as estimated $\pm 1$ std. deviation of the average, for 60 splits.

## 4.3 Censored regression data sets

We will now modify the previous data sets so that they become more challenging. We will consider that they have been *censored*, i.e., values that lie above or below some thresholds have been truncated. This is a realistic setting in the case of physical measurements (e.g., due to the limitation of measuring devices), but clusters of values lying at the end of the range can appear in other cases. In our experiments, we truncated the upper and lower 20% of the previous datasets, while keeping the remaining 60% of data untouched. Note that the methods have no information about the existing truncation or the used thresholds.

As discussed in [1], for this type of data, WGP tries to spread the samples in latent space by using a very sharp warping function and this causes the model problems. Additionally, the computation of the NLPD becomes erroneous due to numerical problems, with some of the tanh functions becoming very close to sign functions. None of these problems were experienced by BWGP, which still works significantly better than a standard GP on this type of problems, see Table 2. The corresponding warping functions are displayed on Figs. 3.(e)-(g).

Table 2: NMSE and NLPD figures for the compared methods on censored data sets.

| Model | MSE | | | NLPD | | |
|---|---|---|---|---|---|---|
| | abalone | creep | ail $(\times 10^{-8})$ | abalone | creep | ailerons |
| GP | $1.27\pm0.12$ | $339.5\pm29.2$ | $1.20\pm0.12$ | $1.54\pm0.05$ | $4.22\pm0.04$ | $-7.70\pm0.05$ |
| BWGP | $1.27\pm0.12$ | $276.8\pm26.8$ | $1.18\pm0.12$ | $0.74\pm0.36$ | $3.68\pm0.17$ | $-7.89\pm0.07$ |
| WGP3 | $1.40\pm0.31$ | $434.6\pm169.0$ | $1.83\pm2.18$ | — | — | — |
| WGP20 | $1.38\pm0.22$ | $382.1\pm93.4$ | $1.39\pm0.78$ | — | — | — |

## 4.4 Classification data sets

Classification can be regarded as an extreme case of censoring or quantisation of a regression data set. We also mentioned in Section 2.2 that the (conditional) generative model of GP classification

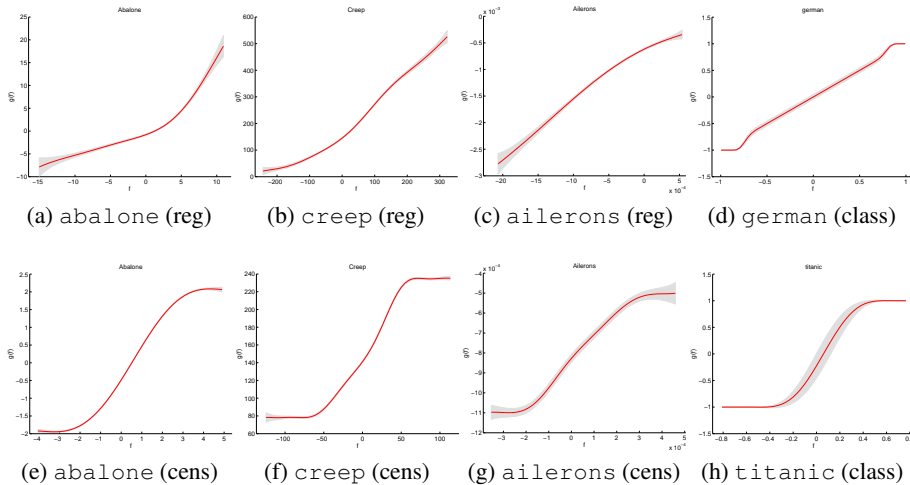

| | | | | | | | | |
|---|---|---|---|---|---|---|---|---|
| (a) `abalone` (reg) | (b) `creep` (reg) | (c) `ailerons` (reg) | (d) `german` (class) | | | | | |
| (e) `abalone` (cens) | (f) `creep` (cens) | (g) `ailerons` (cens) | (h) `titanic` (class) | | | | | |

Figure 3: Inferred warping functions.

Table 3: Error rates (in percentage) for the proposed model on the benchmark from Rätsch [18].

| | ban | bre | dia | fla | ger | hea | ima | rin | spl | thy | tit | two | wav |
|---|---|---|---|---|---|---|---|---|---|---|---|---|---|
| GP | 13.2 | 29.6 | 28.0 | 39.1 | 27.6 | 28.6 | 03.2 | 21.1 | 23.4 | 13.7 | 23.6 | 10.1 | 15.5 |
| BWGP | 10.7 | 29.5 | 24.5 | 33.3 | 23.9 | 23.5 | 02.1 | 04.8 | 17.0 | 04.7 | 22.0 | 04.2 | 12.4 |
| GPC | 10.6 | 29.5 | 24.2 | 33.5 | 24.8 | 21.7 | 02.1 | 07.9 | 22.8 | 04.0 | 22.2 | 04.2 | 11.4 |

could be seen as a particular selection for $g(f)$. So we decided to test the BWGP model on the 13 classification data sets from Rätsch benchmark [18].

Since WGP does not produce any meaningful results on this type of data, as mentioned in [1], we did not include it in the comparison. Instead, we used a standard GP classifier (GPC) using a probit likelihood and expectation propagation for approximate inference. We measured the error rate, which is the performance figure we are interested in for those data sets, averaging over 10 splits of the data. Results from Table 3 show that BWGP is able to match and occasionally exceed the performance of GPC, outperforming in all cases the standard GP. The learned warping functions look similar for the different data sets. We have depicted two typical cases in Figs. 3.(d) and 3.(h). Specially good results are obtained for `german`, `ringnorm`, and `splice`, though we are aware than even better results can be obtained by using an isotropic SE covariance on these data sets [19].

## 5  Discussion and further work

In this work we have shown how it is possible to variationally integrate out the warping function from warped GPs. This is useful to overcome the limitations of maximum likelihood warped GPs, namely: To work in the low data sample regime; to handle censored observations and classification data; to explicitly model output noise; and to allow for warping functions of unlimited flexibility, which may include flat regions. The experiments demonstrate the improved robustness of the BWGP model, which is able to operate properly in a much wider set of scenarios. While a specific model (should it exist) will generally be a better tool for a specific task (e.g., GPC for classification), BWGP behaves as a Swiss Army knife providing good performance on general tasks.

In addition to the tasks discussed in this work, there are other cases in which BWGP can be of immediate application. One example is ordinal regression [8], where the locations and widths of the bins can be integrated out instead of selected. Another potential future application is within the popular field of copulas [20, 21, 22, 23], since they routinely resort to fixed warpings of GPs.

**Acknowledgments**

MLG is grateful to Michalis K. Titsias and the anonymous reviewers for helpful comments.

## Footnotes

[1]We can make $m$, which is the number of inducing inputs and associated inducing variables, as big as we desire (and thus make the sampling arbitrarily dense), independently of the number of available samples $n$.

[2]Note that $\mathbf{C}_{fv}\mathbf{C}_{vv}^{-1}\mathbf{C}_{fv}^\top$ is a Nyström approximation to $\mathbf{C}_{ff}$, whose quality grows with $m$.

[3]Using variational arguments, $q^*(\mathbf{u}) \propto p(\mathbf{u}) \exp(-\frac{1}{2\sigma^2}\mathbf{u}^{\top}\mathbf{C}_{vv}^{-1}\boldsymbol{\Psi}_2\mathbf{C}_{vv}^{-1}\mathbf{u} + \frac{1}{\sigma^2}(\mathbf{y}^{\top}\boldsymbol{\Psi}_1 - \boldsymbol{\psi}_3^{\top})\mathbf{C}_{vv}^{-1}\mathbf{u})$.

[4]Available from `http://www.gatsby.ucl.ac.uk/~snelson/`.

# References

[1] E. Snelson, Z. Ghahramani, and C. Rasmussen. Warped Gaussian processes. In *Advances in Neural Information Processing Systems 16*, 2003.

[2] C. E. Rasmussen. *Evaluation of Gaussian Processes and other Methods for Non-linear Regression*. PhD thesis, University of Toronto, 1996.

[3] M.N. Gibbs. *Bayesian Gaussian Processes for Regression and Classification*. PhD thesis, University of Cambridge, 1997.

[4] C.E. Rasmussen and C.K.I. Williams. *Gaussian Processes for Machine Learning*. Adaptive Computation and Machine Learning. MIT Press, 2006.

[5] M.N. Schmidt. Function factorization using warped gaussian processes. In *Proc. of the 26th International Conference on Machine Learning*, pages 21–928. Omnipress, 2009.

[6] Y. Zhang and D.-Y Yeung. Multi-task warped gaussian process for personalized age estimation. In *IEEE Conf. on Computer Vision and Pattern Recognition*, pages 2622–2629.

[7] C.K.I. Williams and C.E. Rasmussen. Gaussian processes for regression. In *Advances in Neural Information Processing Systems 8*. MIT Press, 1996.

[8] W. Chu and Z. Ghahramani. Gaussian processes for ordinal regression. *Journal of Machine Learning Research*, 6:1019–1041, 2005.

[9] M.K. Titsias and N.D. Lawrence. Bayesian Gaussian process latent variable model. In *Proc. of the 13th International Workshop on Artificial Intelligence and Statistics*, volume 9 of *JMLR: W&CP*, pages 844–851, 2010.

[10] M.K. Titsias. Variational learning of inducing variables in sparse Gaussian processes. In *Proc. of the 12th International Workshop on Artificial Intelligence and Statistics*, 2009.

[11] M. Lázaro-Gredilla and M. Titsias. Variational heteroscedastic Gaussian process regression. In *28th International Conference on Machine Learning (ICML-11)*, pages 841–848, New York, NY, USA, June 2011. ACM.

[12] M. Opper and C. Archambeau. The variational Gaussian approximation revisited. *Neural Computation*, 21(3):786–792, 2009.

[13] M.K. Titsias A.C. Damianou and N.D. Lawrence. Variational gaussian process dynamical systems. In *Advances in Neural Information Processing System 25*. IEEE Conf. publications, 2011.

[14] A. Frank and A. Asuncion. UCI machine learning repository, 2010. `http://archive.ics.uci.edu/ml` University of California, Irvine, School of Information and Computer Sciences.

[15] Materials algorithms project (MAP) program and data library. `http://www.msm.cam.ac.uk/map/map.html`.

[16] D. Cole, C. Martin-Moran, A. G. Sheard, H. K. D. H. Bhadeshia, and D. J. C. MacKay. Modelling creep rupture strength of ferritic steel welds. *Science and Technology of Welding and Joining*, 5:81–90, 2000.

[17] L. Torgo. `http://www.liacc.up.pt/~ltorgo/Regression/`.

[18] G. Rätsch, T. Onoda, and K.-R. Müller. Soft margins for AdaBoost. *Machine Learning*, 42(3):287–320, 2001. `http://people.tuebingen.mpg.de/vipin/www.fml.tuebingen.mpg.de/Members/raetsch/benchmark.1.html`.

[19] A. Naish-Guzman and S. Holden. The generalized FITC approximation. In *Advances in Neural Information Processing Systems 20*, pages 1057–1064. MIT Press, 2008.

[20] R.B. Nelsen. *An Introduction to Copulas*. Springer, 1999.

[21] P.X.-K. Song. Multivariate dispersion models generated from Gaussian copula. *Scandinavian Journal of Statistics*, 27(2):305–320, 2000.

[22] A. Wilson and Z. Ghahramani. Copula processes. In *Advances in Neural Information Processing Systems 23*, pages 2460–2468. MIT Press, 2010.

[23] F.L. Wauthier and M.I. Jordan. Heavy-tailed process priors for selective shrinkage. In *Advances in Neural Information Processing Systems 23*. MIT Press, 2010.

